# Human and Ideal Observers for Detecting Image Curves

**Alan Yuille**
Department of Statistics & Psychology
University of California Los Angeles
Los Angeles CA
yuille@stat.ucla.edu

**Fang Fang**
Psychology, University of Minnesota
Minneapolis MN 55455
fang0057@tc.umn.edu

**Paul Schrater**
Psychology, University of Minnesota
Minneapolis MN 55455
schrater@umn.edu

**Daniel Kersten**
Psychology, University of Minnesota
Minneapolis MN 55455
kersten@umn.edu

## Abstract

This paper compares the ability of human observers to detect target image curves with that of an ideal observer. The target curves are sampled from a generative model which specifies (probabilistically) the geometry and local intensity properties of the curve. The ideal observer performs Bayesian inference on the generative model using MAP estimation. Varying the probability model for the curve geometry enables us investigate whether human performance is best for target curves that obey specific shape statistics, in particular those observed on natural shapes. Experiments are performed with data on both rectangular and hexagonal lattices. Our results show that human observers' performance approaches that of the ideal observer and are, in general, closest to the ideal for conditions where the target curve tends to be straight or similar to natural statistics on curves. This suggests a bias of human observers towards straight curves and natural statistics.

## 1 Introduction

Detecting curves in images is a fundamental visual task which requires combining local intensity cues with prior knowledge about the probable shape of the curve. Curves with strong intensity edges are easy to detect, but those with weak intensity edges can only be found if we have strong prior knowledge of the shape, see figure (1) But, to the best of our knowledge, there have been no experimental studies which test the ability of human observers to perform curve detection for semi-realistic stimuli with locally ambiguous intensity cues or to explore how the difficulty of the task varies with the geometry of the curve.

This paper formulates curve detection as Bayesian inference. Following Geman and Jedynak [6] we define probability distributions $P_G(.)$ for the shape geometry of the target curve and $P_{\text{on}}(.), P_{\text{off}}(.)$ for the intensity on and off the curve. Sampling this model gives us semi-realistic images defined on either rectangular or hexagonal grids. The human ob-

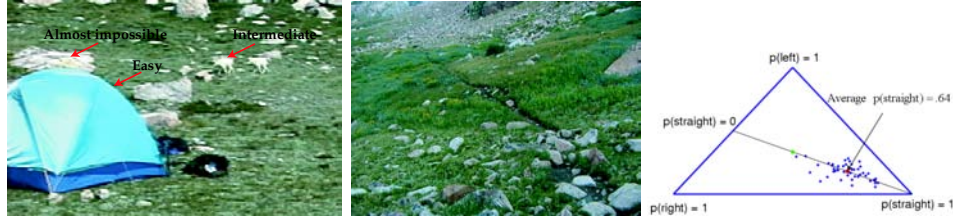

Figure 1: It is plausible that the human visual system is adapted to the shape statistics of curves and paths in images like these. Left panel illustrates the trade-off between the reliability of intensity measurements and priors on curve geometry. The tent is easy to detect because of the large intensity difference between it and the background, so little prior knowledge about its shape is required. But detecting the goat (above the tent) is harder and seems to require prior knowledge about its shape. Centre panel illustrates the experimental task of tracing a curve (or road) in clutter. Right panel shows that the first order shape statistics from 49 object images (one datapoint per image) are clustered round $P(straight) = 0.64$ (with $P(left) = 0.18$ and $P(right) = 0.18$) for both rectangular and hexagonal lattices, see [1].

server's task is to detect the target curve and to report it by tracking it with the (computer) mouse. Human performance is compared with that of an *ideal observer* which computes the target curve using Bayesian inference (implemented by a dynamic programming algorithm). The ideal observer gives a benchmark against which human performance can be measured.

By varying the probability distributions $P_G, P_{on}.P_{off}$ we can explore the ability of the human visual system to detect curves under a variety of conditions. For example, we can vary $P_G$ and determine what changes in $P_{on}.P_{off}$ are required to maintain a pre-specified level of detection performance.

In particular, we can investigate how human performance depends on the geometrical distribution $P_G$ of the curves. It is plausible that the human visual system has adapted to the statistics of the natural world, see figure (1), and in particular to the geometry of salient curves. Our measurements of natural image curves, see figure (1), and studies by [16], [10], [5] and [2], show distributions for shape statistics similar to those found for image intensities statistics [11, 9, 13]. We therefore investigate whether human performance approaches that of the ideal when the probability distributions $P_G$ is similar to that for curves in natural images.

This investigation requires specifying performance measures to determine how close human performance is to the ideal (so that we can quantify whether humans do better or worse *relative to the ideal* for different shape distributions $P_G$). We use two measures of performance. The first is an *effective order parameter* motivated by the order parameter theory for curve detection [14], [15] which shows that the detectability of target curves, by an ideal observer, depends only on an order parameter $K$ which is a function of the probability distributions characterizing the problem. The second measure computes the value of the posterior distribution for the curves detected by the human and the ideal and takes the logarithm of their ratio. (For theoretical reasons this is expected to give a performance measure similar to the effective order parameter).

The experiments are performed by human observers who are required to trace the target curve in the image. We simulated the images first on a rectangle grid and then on a hexagonal grid to test the generality of the results. In these experiments we varied the probability distributions of the geometry $P_G$ and the distribution $P_{on}$ of the intensity on the target curve to allow us to explore a range of different conditions (we kept the distribution $P_{off}$ fixed).

In section (2) we briefly review previous psychophysical studies on edge detection. Sec-

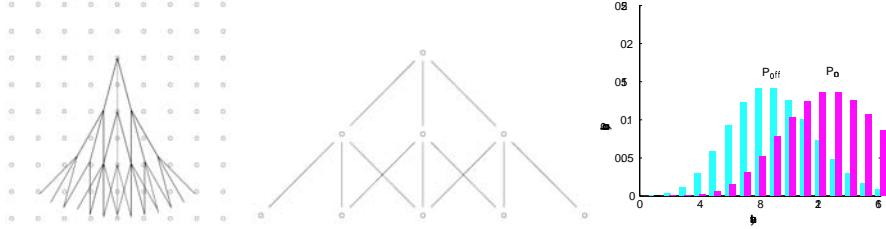

Figure 2: Left panel: the tree structure superimposed on the lattice. Centre panel: a pyramid structure used in the simulations on the rectangular grid. Right panel: Typical distributions of $P_{\text{on}}, P_{\text{off}}$

tion (3) describes our probabilistic model and specifies the ideal observer. In section (4), we describe the order parameter theory and define two performance measures. Sections (5,6) describe experimental results on rectangular and hexagonal grids respectively in terms of our two performance measures.

## 2 Previous Work

Previous psychophysical studies have shown conditions for which the human visual system is able to effectively group contour fragments when embedded in an array of distracting fragments [3, 8]. Most of these studies have focused on the geometrical aspects of the grouping process. For example, it is known that the degree to which a target contour "pops out" depends on the degree of similarity of the orientation of neighboring fragments (typically gabor patches) [3], and that global closure facilitates grouping [8].

Recently, several researchers have shown that psychophysical performance for contour grouping may be understood in terms of the statistical properties of natural contours [12, 5]. For example, Geisler [5] has shown that human contour detection for line segments can be quantitatively predicted from a local grouping rule derived from measurements of local edge statistics.

However, apart from studies that manipulate the contrast of gabor patch tokens [4], there has been little work on how intensity and contour geometry information is combined by the visual system under conditions that begin to approximate those of natural contours. In this paper we attempt to fill this gap by using stimuli sampled from a generative model which enables us to quantitatively characterize the shape and intensity information available for detecting curves and compare human performance with that of an ideal detector.

## 3 The Probabilistic Model for Data Generation

We now describe our model in detail. Following [6], we formulate target curve detection as tree search, see figure (2), through a Q-nary tree. The starting point and initial direction is specified and there are $Q^N$ possible distinct paths down the tree. A target curve hypothesis consists of a set of connected straight-line segments called *segments*. We can represent a path by a sequence of moves $\{t_i\}$ on the tree. Each move $t_i$ belongs to an *alphabet* $\{a_\mu\}$ of size $Q$. For example, the simplest case sets $Q = 3$ with an alphabet $a_1, a_2, a_3$ corresponding to the decisions: (i) $a_1$ – go straight (0 degrees), (ii) $a_2$ – go left (-5 degrees), or (iii) $a_3$ – go right (+ 5 degrees). This determines a path $\mathbf{x}_1, \ldots, \mathbf{x}_N$ in the image lattice where $\mathbf{x}_i, \mathbf{x}_{i+1}$ indicate the start and end points of the $i^{th}$ segment. The relationship between the two representations is given by $\mathbf{x}_{i+1} = \mathbf{x}_i + \mathbf{w}(\mathbf{x}_i - \mathbf{x}_{i-1}, t_i)$, where $\mathbf{w}(\mathbf{x}_i - \mathbf{x}_{i-1}, t_i)$ is a vector of approximately fixed magnitude (choosen to ensure that the segment ends on a pixel) and whose direction depends on the angle of the move $t_i$ relative to the direction of the previous segment $\mathbf{x}_i - \mathbf{x}_{i-1}$. In this paper we restrict $Q = 3$.

We put a prior probability on the geometry of paths down the tree. This is of form $P(\{t_i\}) = \prod_{i=1}^{N} P(t_i)$. We will always require that the probabilities to go left or right are equal and hence we can specify the distribution by the probability $P(straight)$ that the curve goes straight. Our analysis of image curve statistics suggests that $P(straight) = 0.64$ for natural images, see figure (1).

We specify the probability models $P_{on}, P_{off}$ for the image intensity *on* and *off* to be of Poisson form defined over the range $(1, ..., 16)$, see figure (2). This reduced range means that the distributions are expressed as $P_{on}(\mathrm{I} = \mathrm{n}) = (1/K_{on})e^{-\lambda_{on}}\lambda_{on}^{n}/n!$ and $P_{off}(\mathrm{I} = \mathrm{n}) = (1/K_{off})e^{-\lambda_{off}}\lambda_{off}^{n}/n!$, where $K_{on}, K_{off}$ are normalization factors. We fix $\lambda_{off} = 8.0$ and will vary $\lambda_{on}$. The quantity $\lambda_{on} - \lambda_{off}$ is a measure of the local intensity contrast of the target contour and so we informally refer to it as the *signal-to-noise ratio* (SNR).

The Ideal Observer estimates the target curve trajectory by MAP estimation (which we compute using dynamic programming). As described in [6], MAP estimation corresponds to finding the path $\{t_i\}$ with filter measurements $\{y_i\}$ which maximizes the (scaled) log-likelihood ratio, or *reward function*,

$$r(\{t_i\}, \{y_i\}) = \frac{1}{N}\{\log P(Y|X) + \log P(X) - \sum_{i=1}^{N} \log U(t_i)\}$$

$$= \frac{1}{N}\sum_{i=1}^{N}\log\{P_{on}(y_i)/P_{off}(y_i)\} + \frac{1}{N}\sum_{i=1}^{N}\log\{P_G(t_i)/U(t_i)\}, \qquad (1)$$

where $U(.)$ is the uniform distribution (i.e. $U(t) = 1/3 \; \forall t$) and so $\sum_{i=1}^{N}\log U(t_i) = -N\log 3$ which is a constant. The length of the curve is $N = 32$ in our experiments.

We implement this model on both rectangular and hexagonal lattices (the hexagonal lattices equate for contrast at borders, and are visually more realistic). The tree representation used by Geman and Jedynak must be modified when we map onto these lattices. For a rectangular lattice, the easiest way to do this involves defining a *pyramid* where paths start at the apex and the only allowable "moves" are: (i) one step down, (ii) one step down and one step left, and (iii) one step down and one step right. This can be represented by $\mathbf{x}_{i+1} = \mathbf{x}_i + \mathbf{w}(t_i)$ where $t_i \in \{-1, 0, 1\}$ and $\mathbf{w}(-1) = -\vec{i}-\vec{j}$, $\mathbf{w}(0) = -\vec{j}$, $\mathbf{w}(1) = +\vec{i}-\vec{j}$ (where $\vec{i}, \vec{j}$ are the $x, y$ directions on the lattice).

A similar procedure is used on the hexagonal lattice. But for certain geometry probabilities we observed that the sampled curves had "clumping" where the path consists of a large number of zig-zags. This was sometimes confusing to the human observers. So we implemented a higher-order Markov model which explicitly forbade zig-zags. We show experimental results for both the Clumping and No-Cluming models.

To obtain computer simulations of target curves in background clutter we proceed in two stages. In the first stage, we stochastically sample from the distribution $P_G(t)$ to produce a target curve in the pyramid (starting at the apex and moving downwards). In the second stage, we must sample from the likelihood function to generate the image. So if a pixel $\mathbf{x}$ is *on* or *off* the target curve (which we generated in the first stage) then we sample the intensity $I(\mathbf{x})$ from the distribution $P_{on}(I)$ or $P_{off}(I)$ respectively.

## 4 Order Parameters and Performance Measures

Yuille *et al* [14],[15] analyzed the Geman and Jedynak model [6] to determine how the ability to detect the target curve depended on the geometry $P_g$ and the intensity properties $P_{\mathrm{on}}.P_{\mathrm{off}}$. The analysis showed that the ability to detect the target curve behaves as $e^{-KN}$,

where $N$ is the length of the curve and $K$ is an *order parameter*. The larger the value of $K$ then the easier it is to detect the curve.

The order parameter is given by $K = D(P_{on}||P_{off}) + D(P_G||U) - \log Q$ [15], where $U$ is the uniform distribution. If $K > 0$ then detecting the target curve is possible but if $K < 0$ then it becomes impossible to find it (informally, it becomes like looking for a needle in a haystack).

The order parameter illustrates the trade-off between shape and intensity cues and determines which types of curves are easiest to detect by an ideal observer. The intensity cues are quantified by $D(P_{on}||P_{off})$ and the shape cues by $D(P_G||U)$. The easiest curves to detect are those which are straight lines (i.e. $D(P_G||U)$ takes its largest possible value). The hardest curves to detect are those for which the geometry is most random. The stronger the intensity cues (i.e. the bigger $D(P_{on}||P_{off})$) then, of course, the easier the detection becomes.

So when comparing human performance to ideal observers we have to take into account that some types of curves are inherent easier to detect (i.e. thay have larger $K$). Human observers are good at detecting straight line curves but so are ideal obervers. We need performance measures to quantify the *relative effectiveness* of human and ideal observers. Otherwise, we will not be able to conclude that human observers are biased towards particular curve shapes (such as those occuring in natural images).

We now define two performance measures to quantify the relative effectivenes of human and ideal observers. Our first measure is based on the hypothesis that human observers have an "effective order parameter". In other words, their performance on the target curve tracking task behaves like $e^{-NK_H}$ where $K_H$ is an *effective order parameter* which difference from the true order parameter $K$ might reflect a human bias towards straight lines or ecological shape priors. We estimate the effective order parameters by fixing $P_G, P_{\text{off}}$ and adjusting $P_{\text{on}}$ until the observers achieve a fixed performance level of at most 5 errors on a path of length 32. This gives distributions $P_{on}^I, P_{on}^H$ for the ideal and human observers respectively. Then we set $K_H = K - D(P_{on}^H||P_{off}) + D(P_{on}^I||P_{off})$, where $P_{on}^H, P_{on}^I$ are the distributions used by the human and the ideal (respectively) to achieve similar performance.

Our first performance measure is the difference $\Delta K = D(P_{on}^H||P_{off}) - D(P_{on}^I||P_{off})$ between the effective and the true order parameters.

But order parameter analysis should be regarded with caution for the curve detection task used in our experiments. The experimental criterion that the target path be found with 5 or less errors, see section (5), was not included in the theoretical analysis [14],[15]. Also some small corrections need to be made to the order parameters due to the nature of the rectangular grid, see [15] for computer calculations of the size of these corrections. These two effects – the error criterion and the grid correction – means that the order parameters are only approximate for these experimental conditions.

This motivates a second performance measure where we calculate the value of the posterior probability (proportional to the exponential of $r$ in equation (1)) for the curve detected by the human and the ideal observer (for identical distributions $P_G, P_{\text{on}}, P_{\text{off}}$). We measure the *logarithm of the ratio of these values*. (A theoretical relationship can be shown between these two measures).

## 5  Experimental Results on Rectangular Grid

To assess human performance on the road tracking task, we first had a set of 7 observers find the target curve in a tree defined by a rectangular grid figure (3)A. The observer tracked the

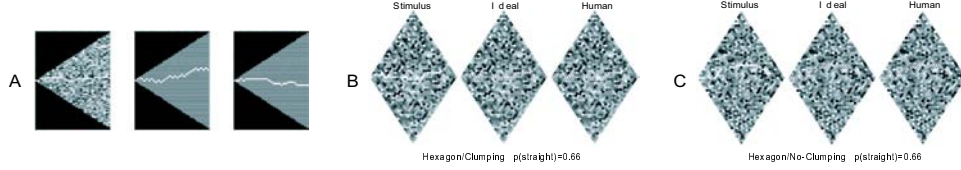

Figure 3: A. Rectangular Grid Stimulus (Left), Example Path: Ideal (Center), Example Path: Human (Right). B & C. Hexagonal Grid Stimulus (Left), Example Path: Ideal (Center), Example Path: Human (Right). Panel C shows an example of a path with higher order constraints to prevent "clumping". There were a number of other differences between the rectangular and hexagonal grid psychophysics, including rectangle samples were slightly smaller than the hexgaons, and feedback was presented to the observers without (rectangular) or with background (hexagonal), and the lowest $p(straight)$ was 0.0 for rectangular and0.1 for hexagonal grids.

contour by starting at the far left corner and making a series of 32 key presses that moved the observer's tracked contour either left, right, or straight at each key press. Each contour estimate was scored by counting the number of positions the observer's contour was off the true path. Each observer had a training period in which the observer was shown examples of contours produced from the four different geometry distributions and practiced tracing in noise.

During an experimental session, the geometry distribution was fixed at one the four possible values and observers were told which geometry distribution was being used to generate the contours. The parameter $\lambda_{on}$ of $P_{on}$ was varied using an adaptive procedure until the human observer managed to repeatedly detect the target curve with at most five misclassified pixels. This gave a *threshold* of $\lambda_{on} - \lambda_{off}$ for each probability distribution defined by $P(straight)$. This threshold could be compared to that of the Ideal Observer (obtained by using dynamic programming to estimate the ideal, also allowing for up to five errors). The process was repeated several times for the four geometry distribution conditions.

The thresholds for 7 observers and the ideal observer are shown in figure 4. These thresholds can be used to calculate our first performance measure ($\Delta K$) and determine how effectively observers are using the available image information at each $P(straight)$.

The results are illustrated in figure (4)B where the human data was averaged over seven subjects. They show that humans perform best for curves with $P(straight) = 0.66$ which is closest to the natural priors, see figure (1). Conversely, $\Delta K$ is biggest for the curves with $P(straight) = 0.0$, which is the condition that differs most from the natural statistics.

We next compute our second performance measure (for which $P_{on}, P_{off}, P_G$ are the same for the ideal and the human observer). The average difference of this performance measure for the each geometry distribution is an alternative way how well observers are using the intensity information as a function of geometry, with a zero difference indicating optimal use of the information. The results are shown in figure (4)C. Notice that the best performance is achieved with $P(straight) = 0.9$.

Observe that the two performance measures give different answers for this experiment. We conclude that our results are consistent either with a bias to ecological statistics or to straight lines. But the rectangular lattice

# 6  Experiments on Hexagonal Lattices

In these experiments we used a hexagonal lattice because, for the human observers, the contrast at the edges corresponding to a left, straight, or right move is the same (in contrast to the rectangular grid, in which left and right moves only share a corner). We also use the same values of $P_{\text{on}}, P_{\text{off}}, P(straight)$ for the humans and the ideal.

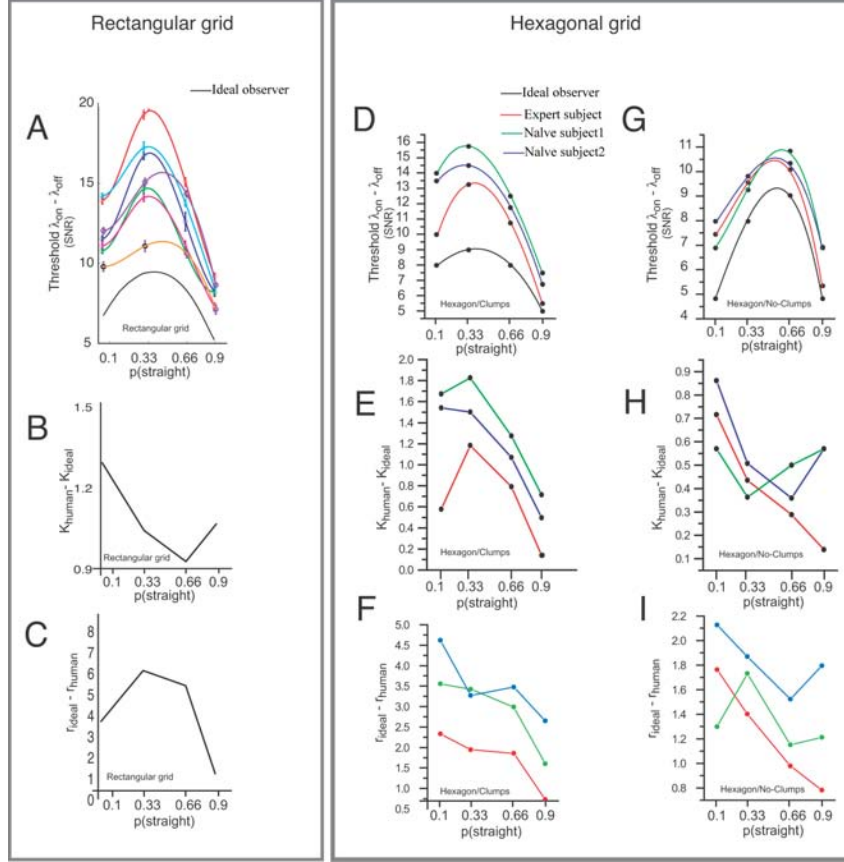

Figure 4: A-C. Psychophysical results on rectangular grid. A. Threshold $\lambda_{on} - \lambda_{off}$ plotted against $P(straight)$. The top seven curves are the results of the seven subjects. The bottom curve is for the ideal observer. B. The difference between human and ideal K order parameters. C. The average reward difference between ideal and human observers. D-I shows psychophyscial results on a hexagonal grid. D-F are for the Clumping condition, and G-I for the No Clumping condition for which high order statistics prevented sharp turns that result in "clumps".

We performed experiments on the hexagonal lattice under four different probabilities for the geometry. These were specified by $P(straight) = 0.10, 0.33, 0.66, 0.90$ (in other words, the straightest curves will be sampled when $P(straight) = 0.90$ and the least straight from $P(straight) = 0.10$). For reasons described previously, we did the experiment in two conditions. (1) allowing zig-zags "Clumping", (2) forbidding zig-zags "No-Clumping". We show examples of the stimuli, the ideal results (indicated by dotted path), and the human results (indicated by dotted path) for the Clumping amd No-Clumping cases in figure (4B & C), respectively.

The threshold SNR results for Clumping and No Clumping are summarized in figures (4D & G. The average $\Delta K = K_{human} - K_{ideal}$ results for Clumping and No Clumping are summarized in figure (4E & H). The average reward difference, $\Delta r = r_{ideal} - r_{human}$, results for Clumping and No Clumping are summarized in figure (4F & I).

Both performance measures give consistent results for the Clumping data suggesting that humans are best when detecting the straightest lines ($P(straight) = 0.9$). But the situation is more complicated for the No Clumping case where human observers show preferences for $P(straight) = 0.9$ or $P(straight) = 0.66$.

# 7 Summary and Conclusions

The results of our experiments suggest that humans are most effective at detecting curves which are straight or which obey ecological statistics. But further experiments are needed to clarify this. Our two performance measures were not always consistent, particularly for the rectangular grid (we are analyzing this discrepency theoretically). The first measure suggested a bias towards ecological statistics on the rectangular grid and for No Clumping stimuli on the hexagonal grid. The second measure showed a bias towards curves with $P(straight) = 0.9$ on the rectangular and hexagonal grids.

To our knowledge, this is the first experiment which tests the performance of human observers for detecting target curves by comparison with that of an ideal observer with ambiguous intensity data. Our novel experimental design and stimuli may cause artifacts due to the rectangular and hexagonal grids. Further experiments may need to "quantize" curves more carefully and reduce the effect of the grids.

Further experiments performed on a larger number of subjects may be able to isolated more precisely the strategy that human observers employ. Do they, for example, make use of a specific geometry prior based on empirical edge statistics [16], [10]. If so, this might account for the bias towards straigthness and natural priors observed in the experiments reported here.

### Acknowledgments

Supported by NIH RO1 EY11507-001, EY02587, EY12691 and, EY013875-01A1, NSF SBR-9631682, 0240148.

# References

[1] Brady, M. J. (1999). Psychophysical investigations of incomplete forms and forms with background. Ph. D., University of Minnesota.

[2] Elder J.H. and Goldberg R.M.. Ecological Statistics of Gestalt Laws for the Perceptual Organization of Contours. *Journal of Vision*, **2**, 324-353. 2002.

[3] Field, D. J., Hayes, A., & Hess, R. F. Contour integration by the human visual system: evidence for a local "association field". *Vision Res*, **33**, (2), 173-93. 1993.

[4] Field, D. J., Hayes, A., & Hess, R. F. The roles of polarity and symmetry in the perceptual grouping of contour fragments. *Spat Vis*, **13**, (1), 51-66.2000.

[5] Geisler W.S. , Perry J.S. , Super B.J. and Gallogly D.P. . Edge co-occurrence in natural images predicts contour grouping performance. *Vision Res*, **41**, (6), 711-24. 2001.

[6] Geman D. and Jedynak B. . "An active testing model for tracking roads from satellite images". *IEEE Trans. Pattern Anal. Mach. Intell.*, **18**, 1-14, 1996.

[7] Hess, R., & Field, D. . Integration of contours: new insights. *Trends Cogn Sci*, **3**, (12), 480-486.1999.

[8] Kovacs, I., & Julesz, B. A closed curve is much more than an incomplete one: effect of closure in figure-ground segmentation. *Proc Natl Acad Sci U S A*, **90**, (16), 7495-7. 1993.

[9] Lee A.B., Huang J.G., and Mumford D.B., "Random collage model for natural images", *Int'l J. of Computer Vision*, Oct. 2000.

[10] Ren X. and Malik J. . "A Probabilistic Multi-scale Model for Contour Completion Based on Image Statistics". In *Proceedings ECCV*. 2002

[11] Ruderman D.L. and Bialek W. , "Statistics of natural images: scaling in the woods", *Phy. Rev. Letter*, 73:814-817, 1994.

[12] Sigman, M., Cecchi, G. A., Gilbert, C. D., & Magnasco, M. O. . On a common circle: natural scenes and Gestalt rules. *Proc Natl Acad Sci U S A*, **98**, (4), 1935-40. 2001.

[13] Wainwright M.J. and Simoncelli E.P., "Scale mixtures of Gaussian and the statistics of natural images", *NIPS*, 855-861, 2000.

[14] Yuille A.L. and Coughlan J.M. . "Fundamental Limits of Bayesian Inference: Order Parameters and Phase Transitions for Road Tracking" . *IEEE PAMI*. Vol. 22. No. 2. February. 2000.

[15] Yuille A.L. , Coughlan J.M., Wu Y-N. and Zhu S.C. . "Order Parameters for Minimax Entropy Distributions: When does high level knowledge help?" *IJCV*. 41(1/2), pp 9-33. 2001.

[16] Zhu S.C. . "Embedding Gestalt Laws in Markov Random Fields – A theory for shape modeling and perceptual organization". *IEEE PAMI*, Vol. 21, No.11, pp1170-1187, Nov, 1999.